# ON TROPISTIC PROCESSING AND ITS APPLICATIONS

Manuel F. Fernández
General Electric Advanced Technology Laboratories
Syracuse, New York  13221

## ABSTRACT

The interaction of a set of tropisms is sufficient in many cases to explain the seemingly complex behavioral responses exhibited by varied classes of biological systems to combinations of stimuli. It can be shown that a straightforward generalization of the tropism phenomenon allows the efficient implementation of effective algorithms which appear to respond "intelligently" to changing environmental conditions. Examples of the utilization of tropistic processing techniques will be presented in this paper in applications entailing simulated behavior synthesis, path-planning, pattern analysis (clustering), and engineering design optimization.

## INTRODUCTION

The goal of this paper is to present an intuitive overview of a general unsupervised procedure for addressing a variety of system control and cost minimization problems. This procedure is based on the idea of utilizing "stimuli" produced by the environment in which the systems are designed to operate as basis for dynamically providing the necessary system parameter updates.

This is by no means a new idea: countless examples of this approach abound in nature, where innate reactions to specific stimuli ("tropisms" or "taxis" --not to be confused with "instincts") provide organisms with built-in first-order control laws for triggering varied responses [8]. (It is hypothesized that "knowledge" obtained through evolution/adaptation or through learning then refines or suppresses most of these primal reactions).

Several examples of the implicit utilization of this approach can also be found in the literature, in applications ranging from behavior modeling to pattern analysis. We very briefly depict some these applications, underlining a common pattern in their formulation and generalizing it through the use of basic field theory concepts and representations. A more rigorous and detailed exposition --regarding both mathematic and application/implementation aspects-- is presently under preparation and should be ready for publication sometime next year ([6]).

## TROPISMS

Tropisms can be defined in general as class-invariant systemic responses to specific sets of stimuli [6]. All time-invariant systems can thus be viewed as tropistic provided that we allow all possible stimuli to form part of our set of inputs. In most tropistic systems, however, response- (or time-) invariance applies only to specific inputs: green plants, for example, twist and grow in the direction of light (phototropism), some birds' flight patterns follow changes in the Earth's magnetic field (magnetotropism), various organisms react to gravitational field

variations (geotropism), etc.

Tropism/stimuli interactions can be portrayed in terms of the superposition of scalar (e.g., potential) or vector (e.g., force) fields exhibiting properties paralleling those of the suitably constrained "reactions" we wish to model [1],[6]. The resulting field can then be used as a basis for assessing the intrinsic cost of pursuing any given path of action, and standard techniques (e.g., gradient-following in the case of scalar fields or divergence computation in the case of vector fields) utilized in determining a response*. In addition, the global view of the situation provided by field representations suggest that a basic theory of tropistic behavior can also be formulated in terms of energy expenditure minimization (Euler-Lagrange equations). This formulation would yield integral-based representations (Feynman path integrals [4],[11]) satisfying the observation that tropistic processes typically obey the principle of least action.

Alternatively, fields may also be collapsed into "attractors" (points of a given "mass" or "charge" in cost space) through laws defining the relationships that are to exist among these "attractors" and the other particles traveling through the space. This provides the simplification that when updating dynamically changing situations only the effects caused by the interaction of the attractors with the particles of interest --rather than the whole cost field-- may have to be recalculated.

For example, appropriately positioned point charges exerting on each other an electrostatic force inversely proportional to the square of their distance can be used to represent the effects of a coulombic-type cost potential field. A particle traveling through this field would now be affected by the combination of forces ensuing from the interaction of the attractors' charges with its own. If this particle were then to passively follow the composite of the effects of these forces it would be following the gradient of the cost field (i.e., the vector resulting from the superposition of the forces acting on the particle would point in the direction of steepest change in potential).

Finally, other representations of tropism/stimuli interactions (e.g., Value-Driven Decision Theory approaches) entail associating "profit" functions (usually sigmoidal) with each tropism, modeling the relative desirability of triggering a reaction as a function of the time since it was last activated [9]. These representations are

---

* In order to bring extra insight into tropism/stimuli interactions and simplify their formulation, one may exchange vector and scalar field representations through the utilization of appropriately selected mappings. Some of the most important of such mappings are the gradient operator (particularly so because the gradient of a scalar --potential-- field is proportional to a "force" --vector-- field), the divergence (which may be thought of as performing in vector fields a function analogous to that performed in scalar fields by the gradient), and their combinations (e.g., the Laplacian, a scalar-to-scalar mapping which can be visualized as performing on potential fields the equivalent of a second derivative operation.

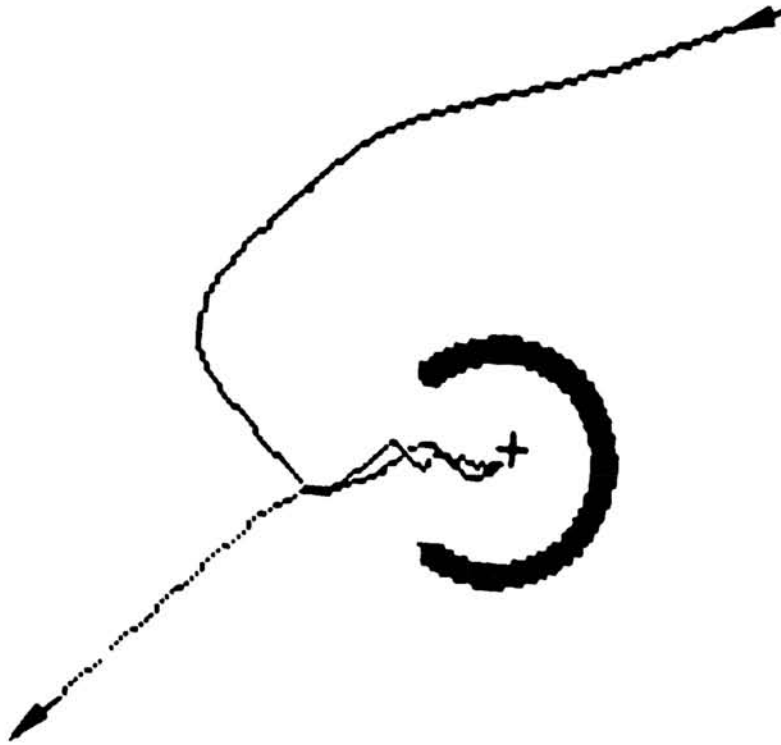

- *Model fly as a positive geotropistic point of mass M.*
- *Model fence stakes as negative geotropistic points with masses $m_1, m_2, \ldots, m_N$.*
- *At each update time compute sum of forces acting on frog:*

$$F = k \left( \frac{M}{d_M^2} - \sum_{i=1}^{N} \frac{m_i}{d_{m_i}^2} \right) m,$$

- *Compute frog's heading and acceleration based on the ensuing force; then update frog's position.*

Figure 1: Attractor-based representation of a frog-fence-fly scenario (see [1] for a vector-field representation). The objective is to model a frog's path-planning decision-making process when approaching a fly in the presence of obstacles. (The picket fence is represented by the elliptical outline with an opening in the back, the fly --inside the fenced space-- is represented by a "+" sign, and arrows are used to indicate the direction of a frog's trajectory into and out of fenced area).

particularly amenable to neural-net implementations [6].

TROPISTIC PROCESSING

Tropistic processing entails building into systems tropisms appropriate for the environment in which these systems are expected to operate. This allows taking advantage of environment-produced "stimuli" for providing the required control for the systems' behavior.

The idea of tropistic processing has been utilized with good results in a variety of applications. Arbib et.al., for example, have implicitly utilized tropistic processing to describe a batrachian's reaction to its environment in terms of what may be visualized as magnetic (vector) fields' interactions [1].

Watanabe [12] devised for pattern analysis purposes an interaction of tropisms ("geotropisms") in which pattern "atoms" are attracted to each other, and hence "clustered", subject to a squared-inverse-distance ("feature distance") law similiar to that from gravitational mechanics. It can be seen that if each pattern atom were considered an "organism", its behavior would not be conceptually different from that exhibited by Arbibian frogs: in both cases organisms passively follow the force vectors resulting from the interaction of the environmental stimuli with the organisms' tropisms. It is interesting, though, to note that the "organisms'" behavior will nonetheless appear "intelligent" to the casual observer.

The ability of tropistic processes to emulate seemingly rational behavior is now begining to be explored and utilized in the development of synthetic-psychological models and experiments. Braitenberg, for example, has placed tropisms as the primal building block from which his models for cognition, reason, and emotions evolve [3]**; Barto [2] has suggested the possibility of combining tropisms and associative (reinforced) learning, with aims at enabling the automatic triggering of behavioral responses by previously experienced situations; and Fernández [6] has used CROBOTS [10], a virtual multiprocessor emulator, as laboratory for evaluating the effects of modifying tropistic responses on the basis of their projected future consequences.

Other applications of tropistic processing presently being investigated include path-planning and engineering design optimization [6]. For example, consider an air-reconnaissance mission deep behind enemy lines; as the mission progresses and unexpected SAM sites are discovered, contingency flight paths may be developed in real time simply by modeling each SAM or interdiction site as a mass point towards which the aircraft exhibits negative geotropistic tendencies (i.e., gravitational forces repel it), and modeling the objective as a positive geotropistic point. A path to

** Of particular interest within the sole context of Tropistic Processing is Dewdney's [5] commented version of the first chapters of Braitenberg's book [3], in which the "behavior" of mechanically very simple cars, provided with "eyes" and phototropism-supporting connections (including Ledley-type "neurons" [4]), is "analyzed".

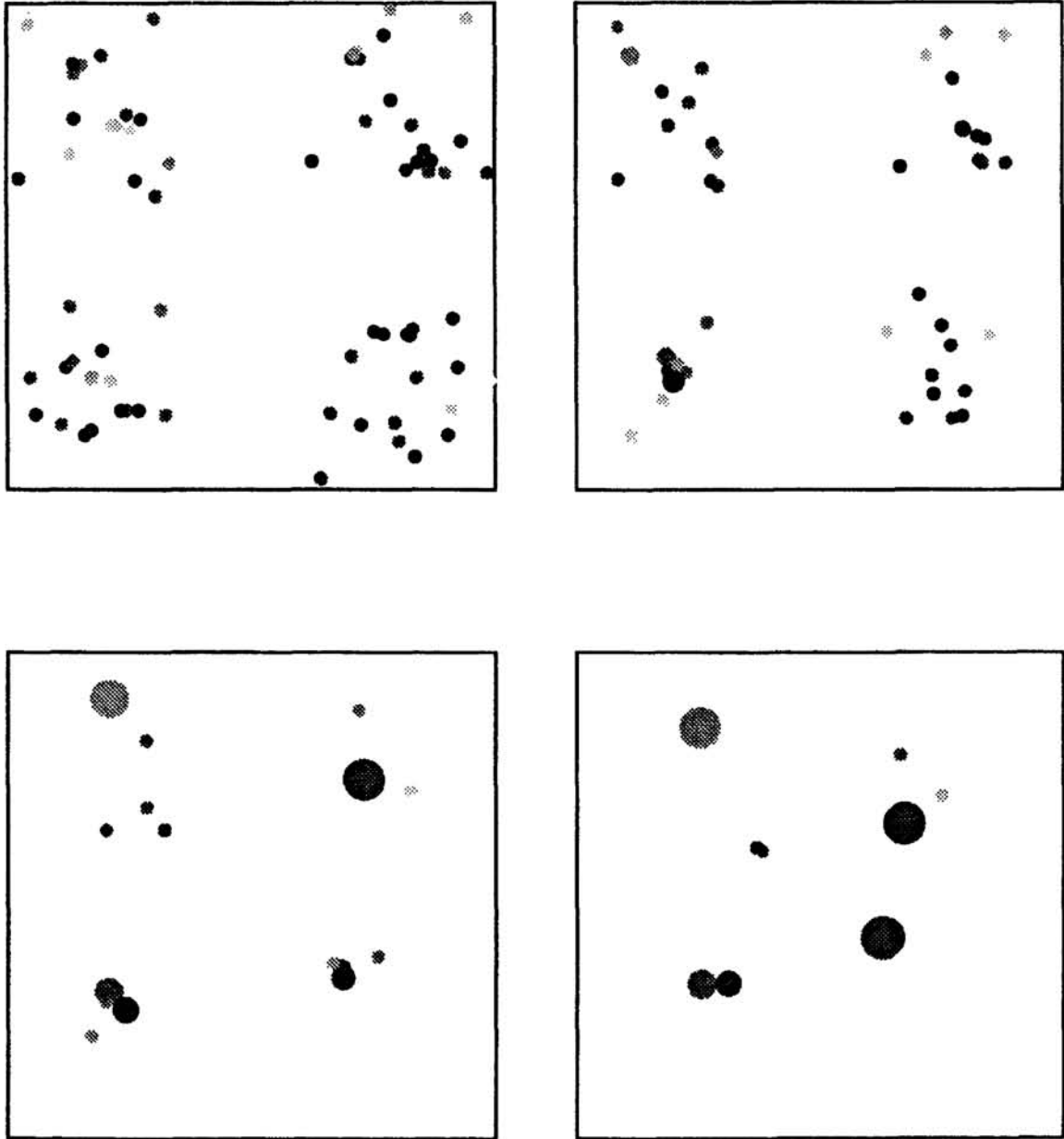

Figure 2 (Geotropistic clustering [12]): The problem being portrayed
here is that of clustering dots distributed in [x,y]-space as shown
and uniformly in color ([red,blue,green]). The approach followed is
that outlined in Figure 1, with the differences that normalized
(Mahalanobis) distances are used and when merges occur, conservation
of momentum is observed. Tags are also kept --specifying with which
dots and in what order merges occur-- to allow drawing cluster
boundaries in the original data set. (Efficient implementation of
this clustering technique entails using a ring of processors, each
of which is assigned the "features" of one or more "dots" and the
task of carrying out computations with respect to these features. If
the features of each dot are then transmitted through the ring, all
the forces imposed on it by the rest will have been determined upon
completion of the circuit).

the target will then be automatically drawn by the interaction of the tropisms with the gravitational forces. (Once the mission has been completed, the target and its effects can be eliminated, leaving active only the repulsive forces, which will then "guide" the airplane out of the danger zone).

In engineering design applications such as lens modeling and design, lenses (gradient-index type, for example) can be modeled in terms of photons attempting to reach an objective plane through a three-dimensional scalar field of refraction indices; modeling the process tropistically (in a manner analogous to that of the air-reconnaissance example above) would yield the least-action paths that the individual photons would follow. Similarly, in "surface-of-revolution" fuselage design ("Newton's Problem"), the characteristics of the interaction of forces acting within a sheet of metal foil when external forces (collisions with a fluid's molecules) are applied can be modeled in terms of tropistic reactions which will tend to reconfigure the sheet so as to make it present the least resistance to friction when traversing a fluid.

Additional applications of tropistic processing include target tracking and multisensor fusion (both can be considered instances of "clustering") [6], resource allocation and game theory (both closely related to path-planning) [9], and an assortment of other cost-minimization functions. Overall, however, one of the most important applications of tropistic processing may be in the modeling and understanding of analog processes [6], the imitation of which may in turn lead to the development of effective strategies

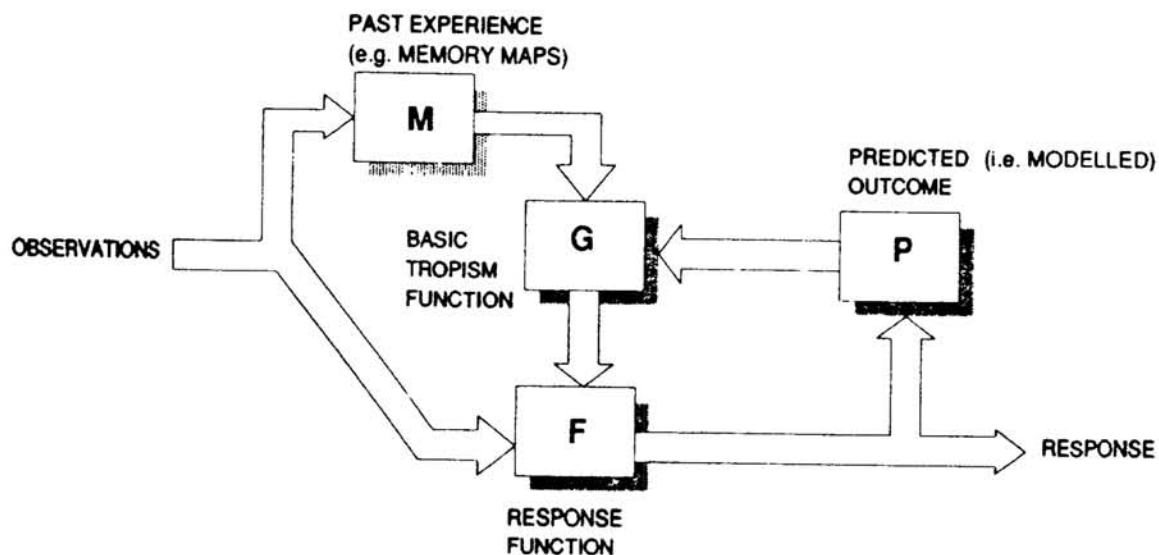

**TROPISM-BASED SYSTEM**

Figure 3: The combination of tropisms and associative (reinforced) learning can be used to enable the automatic triggering of behavioral responses by previously experienced situations [2]. Also, the modeled projection of the future consequences of a tropistic decision can be utilized in the modification of such decision [6]. (Note analogy to filtering problem in which past history and predicted behavior are used to smooth present observations).

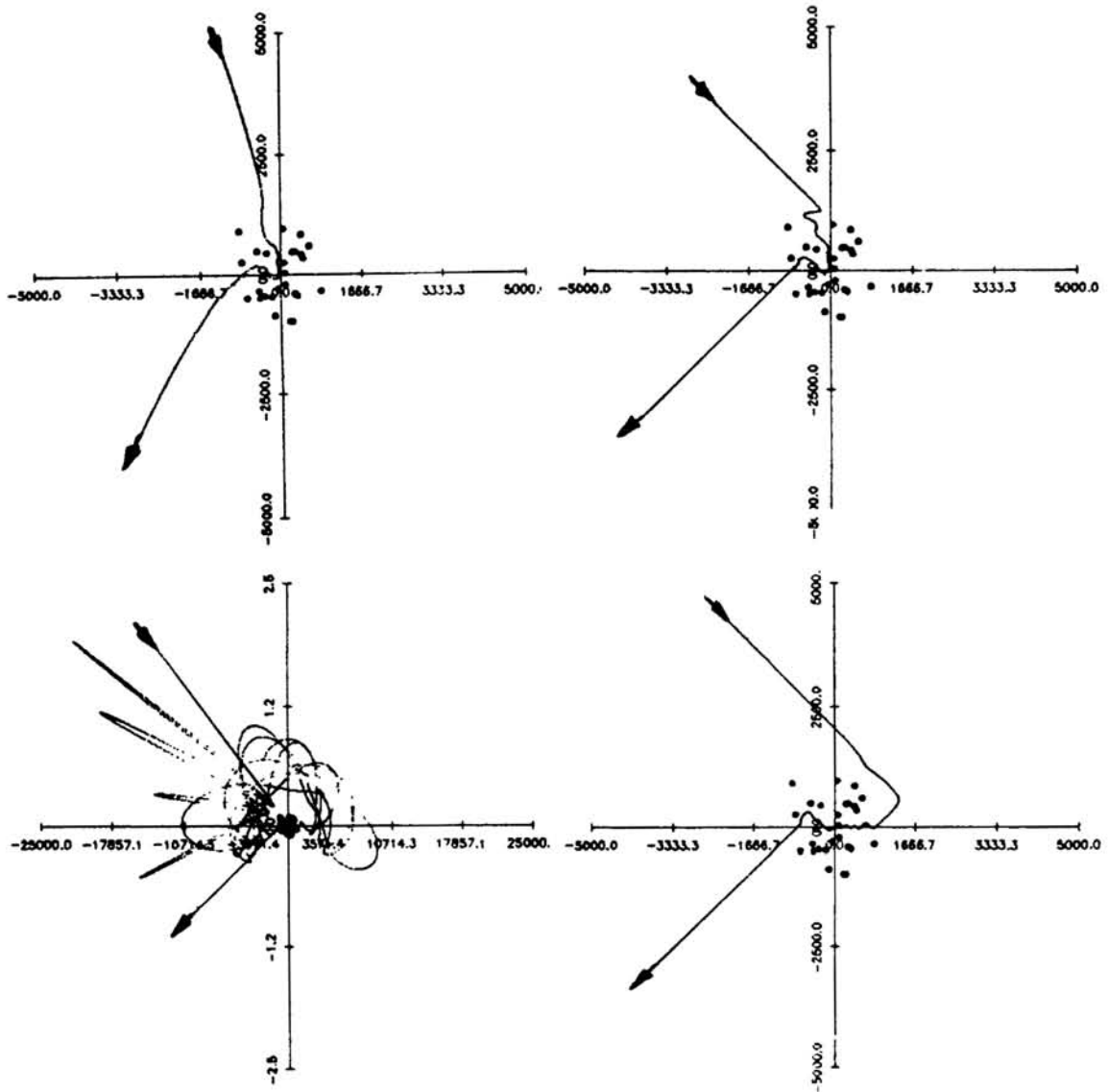

Figure 4: Simplified representation of air-reconnaissance mission
example (see text): objective is at center of coordinate axis, thick
dots represent SAM sites, and arrows denote airplane's direction of
flight (airplane's maximum attainable speed and acceleration are
constrained). All portrayed scenarios are identical except for
tropistic control-law parameters (mainly objective to SAM-sites mass
ratios in the first three scenarios). Varying the masses of the
objective and SAM sites can be interpreted as trading off the
relative importance of the mission vs. the aircraft's safety, and
can produce dramatically differing flight paths, induce chaotic
behavior (bottom-left scenario), or render the system unstable. The
bottom-right scenario portrays the situation in which a tropistic
decision is projected into the future and, if not meeting some
criterion, modified (altering the direction of flight --e.g.,
following an isokline--, re-evaluating the mission's relative
importance --revising masses--, changing the update rate, etc.).

for taking full advantage of parallel architectures [11]***. It is thus expected that the flexibility of tropistic processes to adapt to changing environmental conditions will prove highly valuable to the advancement of areas such as robotics, parallel processing and artificial intelligence, where at the very least they will provide some decision-making capabilities whenever unforeseen circumstances are encountered.

## ACKNOWLEDGEMENTS

Special thanks to D. P. Bray for the ideas provided in our many discussions and for the development of the finely detailed simulations that have enabled the visualization of unexpected aspects of our work.

## REFERENCES

[1] Arbib, M.A. and House, D.H.: "Depth and Detours: Decision Making in Parallel Systems". IEEE Workshop on Languages for Automation: Cognitive Aspects in Information Processing; pp. 172-180 (1985).

[2] Barto, A.G. (Editor): "Simulation Experiments with Goal-Seeking Adaptive Elements". Avionics Laboratory, Wright-Patterson Air Force Base, OH. Report # AFWAL-TR-84-1022. (1984).

[3] Braitenberg, V.: Vehicles: Experiments in Synthetic Psychology. The MIT Press. (1984).

[4] Cheng, G.C.; Ledley, R.S.; and Ouyang, B.: "Pattern Recognition with Time Interval Modulation Information Coding". IEEE Transactions on Aerospace and Electronic Systems. AES-6, No.2; pp. 221-227 (1970).

[5] Dewdney, A.K.: "Computer Recreations". Scientific American. Vol.256, No.3; pp. 16-26 (1987).

[6] Fernández, M.F.: "Tropistic Processing". To be published (1988).

[7] Feynman, R.P.: Statistical Mechanics: A Set of Lectures. Frontiers in Physics Lecture Note Series (1982).

[8] Hirsch, J.: "Nonadaptive Tropisms and the Evolution of Behavior". Annals of the New York Academy of Sciences. Vol.223; pp. 84-88 (1973).

[9] Lucas, G. and Pugh, G.: "Applications of Value-Driven Automation Methodology for the Control and Coordination of Netted Sensors in Advanced C**3". Report # RADC-TR-80-223. Rome Air Development Center, NY. (1980).

[10] Poindexter, T.: "CROBOTS". Manual, programs, and files (1985). 2903 Winchester Dr., Bloomington, IL., 61701.

[11] Wallqvist, A.; Berne, B.J.; and Pangali, C.: "Exploiting Physical Parallelism Using Supercomputers: Two Examples from Chemical Physics". Computer. Vol.20, No.5; pp. 9-21 (1987).

[12] Watanabe, S.: Pattern Recognition: Human and Mechanical. John Wiley & Sons; pp. 160-168 (1985).

---

*** Optical Fourier transform operations, for instance, can be modeled in high-granularity machines through a procedure analogous to the gradient-index lens simulation example, with processors representing diffraction-grating "atoms" [6].